# Improving on Expectation Propagation

**Manfred Opper**
Computer Science, TU Berlin
opperm@cs.tu-berlin.de

**Ulrich Paquet**
Computer Laboratory, University of Cambridge
ulrich@cantab.net

**Ole Winther**
Informatics and Mathematical Modelling, Technical University of Denmark
owi@imm.dtu.dk

## Abstract

A series of corrections is developed for the fixed points of Expectation Propagation (EP), which is one of the most popular methods for approximate probabilistic inference. These corrections can lead to improvements of the inference approximation or serve as a sanity check, indicating when EP yields unrealiable results.

## 1 Introduction

The *expectation propagation* (EP) message passing algorithm is often considered as the method of choice for approximate Bayesian inference when both good accuracy and computational efficiency are required [5]. One recent example is a comparison of EP with extensive MCMC simulations for Gaussian process (GP) classifiers [4], which has shown that not only the predictive distribution, but also the typically much harder marginal likelihood (the partition function) of the data, are approximated remarkably well for a variety of data sets. However, while such empirical studies hold great value, they can not guarantee the same performance on other data sets or when completely different types of Bayesian models are considered.

In this paper methods are developed to assess the quality of the EP approximation. We compute explicit expressions for the remainder terms of the approximation. This leads to various corrections for partition functions and posterior distributions. Under the hypothesis that the EP approximation works well, we identify quantities which can be assumed to be small and can be used in a series expansion of the corrections with increasing complexity. The computation of low order corrections in this expansion is often feasible, typically require only moderate computational efforts, and can lead to an improvement to the EP approximation or to the indication that the approximation cannot be trusted.

## 2 Expectation Propagation in a Nutshell

Since it is the goal of this paper to compute corrections to the EP approximation, we will not discuss details of EP *algorithms* but rather characterise the fixed points which are reached when such algorithms converge.

EP is applied to probabilistic models with an unobserved latent variable $\mathbf{x}$ having an intractable distribution $p(\mathbf{x})$. In applications $p(\mathbf{x})$ is usually the Bayesian posterior distribution conditioned on a set of observations. Since the dependency on the latter variables is not important for the subsequent theory, we will skip them in our notation.

It is assumed that $p(\mathbf{x})$ factorizes into a product of *terms* $f_n$ such that

$$p(\mathbf{x}) = \frac{1}{Z} \prod_n f_n(\mathbf{x}) \,, \tag{1}$$

where the normalising partition function $Z = \int d\mathbf{x} \prod_n f_n(\mathbf{x})$ is also intractable. We then assume an approximation to $p(\mathbf{x})$ in the form

$$q(\mathbf{x}) = \prod_n g_n(\mathbf{x}) \tag{2}$$

where the terms $g_n(\mathbf{x})$ belong to a tractable, e.g. exponential family of distributions. To compute the optimal parameters of the $g_n$ term approximation a set of auxiliary *tilted* distributions is defined via

$$q_n(\mathbf{x}) = \frac{1}{Z_n} \left( \frac{q(\mathbf{x}) f_n(\mathbf{x})}{g_n(\mathbf{x})} \right) \,. \tag{3}$$

Here a *single* approximating term $g_n$ is replaced by an original term $f_n$. Assuming that this replacement leaves $q_n$ still tractable, the parameters in $g_n$ are determined by the condition that $q(\mathbf{x})$ and all $q_n(\mathbf{x})$ should be made as similar as possible. This is usually achieved by requiring that these distributions share a set of generalised moments (which usually coincide with the sufficient statistics of the exponential family). Note, that we will *not* assume that this *expectation consistency* [8] for the moments is derived by minimising a Kullback–Leibler divergence, as was done in the original derivations of EP [5]. Such an assumption would limit the applicability of the approximate inference and exclude e.g. the approximation of models with binary, Ising variables by a Gaussian model as in one of the applications in the last section.

The corresponding approximation to the normalising partition function in (1) was given in [8] and [7] and reads in our present notation[1]

$$Z_{EP} = \prod_n Z_n \,. \tag{4}$$

## 3   Corrections to EP

An expression for the remainder terms which are neglected by the EP approximation can be obtained by solving for $f_n$ in (3), and taking the product to get

$$\prod_n f_n(\mathbf{x}) = \prod_n \left( \frac{Z_n q_n(\mathbf{x}) g_n(\mathbf{x})}{q(\mathbf{x})} \right) = Z_{EP} \, q(\mathbf{x}) \prod_n \left( \frac{q_n(\mathbf{x})}{q(\mathbf{x})} \right) \,. \tag{5}$$

Hence $Z = \int d\mathbf{x} \prod_n f_n(\mathbf{x}) = Z_{EP} R$, with

$$R = \int d\mathbf{x} \, q(\mathbf{x}) \prod_n \left( \frac{q_n(\mathbf{x})}{q(\mathbf{x})} \right) \quad \text{and} \quad p(\mathbf{x}) = \frac{1}{R} \, q(\mathbf{x}) \prod_n \left( \frac{q_n(\mathbf{x})}{q(\mathbf{x})} \right) \,. \tag{6}$$

This shows that corrections to EP are small when all distributions $q_n$ are indeed close to $q$, justifying the optimality criterion of EP. For related expansions, see [2, 3, 9].

Exact probabilistic inference with the corrections described here again leads to intractable computations. However, we can derive exact *perturbation expansions* involving a series of corrections with increasing computational complexity. Assuming that EP already yields a good approximation, the computation of a small number of these terms maybe sufficient to obtain the most dominant corrections. On the other hand, when the leading corrections come out large or do not sufficiently decrease with order, this may indicate that the EP approximation is inaccurate. Two such perturbation expansions are be presented in this section.

## 3.1 Expansion I: Clusters

The most basic expansion is based on the variables $\varepsilon_n(\mathbf{x}) = \frac{q_n(\mathbf{x})}{q(\mathbf{x})} - 1$ which we can assume to be typically small, when the EP approximation is good. Expanding the products in (6) we obtain the correction to the partition function

$$R = \int d\mathbf{x}\, q(\mathbf{x}) \prod_n \left(1 + \varepsilon_n(\mathbf{x})\right) \tag{7}$$

$$= 1 + \sum_{n_1 < n_2} \left\langle \varepsilon_{n_1}(\mathbf{x})\varepsilon_{n_2}(\mathbf{x}) \right\rangle_q + \sum_{n_1 < n_2 < n_3} \left\langle \varepsilon_{n_1}(\mathbf{x})\varepsilon_{n_2}(\mathbf{x})\varepsilon_{n_3}(\mathbf{x}) \right\rangle_q + \dots, \tag{8}$$

which is a finite series in terms of growing clusters of "interacting" variables $\varepsilon_n(\mathbf{x})$. Here the brackets $\langle \dots \rangle_q$ denote expectations with respect to the distribution $q$. Note, that the first order term $\sum_n \langle \varepsilon_n(\mathbf{x}) \rangle_q = 0$ vanishes by the normalization of $q_n$ and $q$. As we will see later, the computation of corrections is feasible when $q_n$ is just a finite *mixture* of $K$ simpler densities from the exponential family to which $q$ belongs. Then the number of mixture components in the $j$-th term of the expansion of $R$ is just of the order $\mathcal{O}(K^j)$ and an evaluation of low order terms should be tractable.

In a similar way, we get

$$p(\mathbf{x}) = \frac{q(\mathbf{x}) \left(1 + \sum_n \varepsilon_n(\mathbf{x}) + \sum_{n_1 < n_2} \varepsilon_{n_1}(\mathbf{x})\varepsilon_{n_2}(\mathbf{x}) + \dots\right)}{1 + \sum_{n_1 < n_2} \left\langle \varepsilon_{n_1}(\mathbf{x})\varepsilon_{n_2}(\mathbf{x}) \right\rangle_q + \dots}, \tag{9}$$

In order to keep the resulting density normalized to one, we should keep as many terms in the numerator as in the denominator. As an example, the first order correction to $q(\mathbf{x})$ is

$$p(\mathbf{x}) \approx \sum_n q_n(\mathbf{x}) - (N-1)q(\mathbf{x}). \tag{10}$$

## 3.2 Expansion II: Cumulants

One of most important applications of EP is to the case of statistical models with Gaussian process priors. Here $\mathbf{x}$ is a latent variable with Gaussian prior distribution and covariance $\mathbb{E}[\mathbf{x}\mathbf{x}^\top] = \mathbf{K}$ where $\mathbf{K}$ is the kernel matrix. In this case we have $N+1$ terms $f_0, f_1, \dots, f_N$ in (1) where $f_0(\mathbf{x}) = g_0(\mathbf{x}) = \exp[-\frac{1}{2}\mathbf{x}^\top \mathbf{K}^{-1}\mathbf{x}]$. For $n \geq 1$ each $f_n(\mathbf{x}) = t_n(x_n)$ is the *likelihood term* for the $n^{\text{th}}$ observation which depends only on a single component $x_n$ of the vector $\mathbf{x}$.

The corresponding approximating terms are chosen to be Gaussian of the form $g_n(x) \propto e^{\gamma_n x - \frac{1}{2}\lambda_n x^2}$. The $2N$ parameters $\gamma_n$ and $\lambda_n$ are determined in such a way that $q(\mathbf{x})$ and the distributions $q_n(\mathbf{x})$ have the same first and second *marginal moments* $\langle x_n \rangle$ and $\langle x_n^2 \rangle$. In this case, the computation of corrections (7) would require the computation of multivariate integrals of increasing dimensionality. Hence, a different type of expansion seems more appropriate. The main idea is to expand with respect to the higher order cumulants of the distributions $q_n$.

To derive this expansion, we simplify (6) using the fact that $q(\mathbf{x}) = q(\mathbf{x}_{\backslash n}|x_n)q(x_n)$ and $q_n(\mathbf{x}) = q(\mathbf{x}_{\backslash n}|x_n)q_n(x_n)$, where we have (with a slight abuse of notation) introduced $q(x_n)$ and $q_n(x_n)$, the marginals of $q(\mathbf{x})$ and $q_n(\mathbf{x})$. Thus $p(\mathbf{x}) = \frac{1}{R}\, q(\mathbf{x})F(\mathbf{x})$ and $R = \int d\mathbf{x}\, q(\mathbf{x})F(\mathbf{x})$, where

$$F(\mathbf{x}) = \prod_n \left(\frac{q_n(x_n)}{q(x_n)}\right). \tag{11}$$

Since $q(x_n)$ and the $q_n(x_n)$ have the same first two cumulants, corrections can be expressed by the *higher cumulants* of the $q_n(x_n)$ (note, that the higher cumulants of $q(x_n)$ vanish). The cumulants $c_{ln}$ of $q_n(x_n)$ are defined by their *characteristic functions* $\chi_n(k)$ via

$$q_n(x_n) = \int \frac{dk}{2\pi}\, e^{-ikx_n}\chi_n(k) \quad \text{and} \quad \ln\chi_n(k) = \sum_l (i)^l \frac{c_{ln}}{l!}\, k^l. \tag{12}$$

Expressing the Gaussian marginals $q(x_n)$ by their first and second cumulants, the means $m_n$ and the variances $S_{nn}$ and introducing the function

$$r_n(k) = \sum_{l \geq 3} (i)^l \frac{c_{ln}}{l!}\, k^l \tag{13}$$

which contains the contributions of all higher order cumulants, we get

$$F(\mathbf{x}) = \prod_n \left( \frac{\int dk_n \ \exp\left[-ik_n(x_n - m_n) - \frac{1}{2}S_{nn}k_n^2 + r_n(k_n)\right]}{\int dk_n \ \exp\left[-ik_n(x_n - m_n) - \frac{1}{2}S_{nn}k_n^2\right]} \right) \tag{14}$$

$$= \int d\boldsymbol{\eta} \sqrt{\prod_n \frac{S_{nn}}{2\pi}} \ \exp\left[ -\sum_n \frac{S_{nn}\eta_n^2}{2} \right] \exp\left[ \sum_n r_n \left( \eta_n - i\frac{(x_n - m_n)}{S_{nn}} \right) \right] \tag{15}$$

where in the last equality we have introduced a shift of variables $\eta_n = k_n + i\frac{(x_n - m_n)}{S_{nn}}$.

An expansion can be performed with respect to the cumulants in the terms $g_n$ which had been neglected in the EP approximation. The basic computations are most easily explained for the correction $R$ to the partition function.

### 3.2.1 Correction to the partition function

Since $q(\mathbf{x})$ is a multivariate Gaussian of the form $q(\mathbf{x}) = \mathcal{N}(\mathbf{x}; \mathbf{m}, \mathbf{S})$, the correction $R$ to the partition $Z$ involves a double Gaussian average over the vector $\mathbf{x}$ and the set of $\eta_n$. This can be simplified by combining them into a *single complex* zero mean Gaussian random vector defined as $z_n = \eta_n - i\frac{x_n - m_n}{S_{nn}}$ such that

$$R = \left\langle \exp\left[ \sum_n r_n(z_n) \right] \right\rangle_{\mathbf{z}} \tag{16}$$

The most remarkable property of the Gaussian $\mathbf{z}$ is its covariance which is easily found to be

$$\langle z_i z_j \rangle_{\mathbf{z}} = -\frac{S_{ij}}{S_{ii}S_{jj}} \quad \text{when } i \neq j, \quad \text{and} \quad \langle z_i^2 \rangle_{\mathbf{z}} = 0 \ . \tag{17}$$

The last equation has important consequences for the surviving terms in an expansion of $R$!

Assuming that the $g_n$ are small we perform a power series expansion of $\ln R$

$$\ln R = \ln \left\langle \exp\left[ \sum_n r_n(z_n) \right] \right\rangle_{\mathbf{z}} = \sum_n \langle r_n \rangle_{\mathbf{z}} + \frac{1}{2}\left\langle \left( \sum_n r_n \right)^2 \right\rangle_{\mathbf{z}} - \frac{1}{2}\left( \sum_n \langle r_n \rangle_{\mathbf{z}} \right)^2 \pm \dots \tag{18}$$

$$= \frac{1}{2} \sum_{m \neq n} \langle r_m r_n \rangle_{\mathbf{z}} \pm \dots \qquad = \sum_{m \neq n} \sum_{l \geq 3} \frac{c_{ln}c_{lm}}{l!} \left( \frac{S_{nm}}{S_{nn}S_{mm}} \right)^l \pm \dots \tag{19}$$

Here we have repeatedly used the fact that each factor $z_n$ in expectations $\langle z_n^l z_m^s \rangle$ have to be paired (by Wick's theorem) with a factor $z_m$ where $m \neq n$ (diagonal terms vanish by (17)). This gives nonzero contributions only, when $l = s$ and there are $l!$ ways for pairing.[2]

This expansion gives a hint why EP may work typically well for multivariate models when covariances $S_{ij}$ are small compared to the variances $S_{ii}$. While we may expect that $\ln Z_{EP} = \mathcal{O}(N)$ where $N$ is the number of variables $x_n$, the vanishing of the "self interactions" indicates that corrections may not scale with $N$.

### 3.2.2 Correction to marginal moments

The predictive density of a novel observation can be treated by extending the Gaussian prior to include a new latent variable $x_*$ with $\mathbb{E}[x_*\mathbf{x}] = \mathbf{k}_*$ and $\mathbb{E}[x_*^2] = k_*$, and appears as an average of a likelihood term over the posterior marginal of $x_*$.

A correction for the predictive density can also be derived in terms of the cumulant expansion by averaging the conditional distribution $p(x_*|\mathbf{x}) = \mathcal{N}(x_*; \mathbf{k}_*^\top \mathbf{K}^{-1}\mathbf{x}, \sigma_*^2)$ with $\sigma_*^2 = k_* - \mathbf{k}_*^\top \mathbf{K}^{-1}\mathbf{k}_*$. Using the expression (15) we obtain (where we set $R = 1$ in (6) to lowest order)

$$p(x_*) = \int d\mathbf{x} \ p(x_*|\mathbf{x}) \ p(\mathbf{x}) = \mathcal{N}(x_*; \mu_{x_*}, s_{x_*}^2) \left\langle 1 + \sum_n r_n \left( \eta_n - i\frac{x_n - m_n}{S_{nn}} \right) + \dots \right\rangle_{\boldsymbol{\eta}, \mathbf{x} \sim \mathcal{N}(\mathbf{x}; \boldsymbol{\mu}, \boldsymbol{\Sigma})} \tag{20}$$

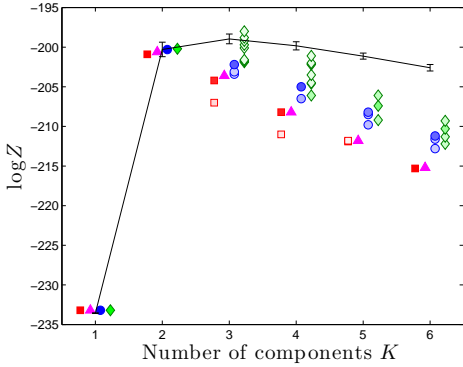

Figure 1: $\ln Z$ approximations obtained from $q(\mathbf{x})$'s factorization in (2), for sec. 4.1's mixture model, as obtained by: variational Bayes (see [1] for details) as red squares; $\alpha = \frac{1}{2}$ in Minka's $\alpha$-divergence message passing scheme, described in [6], as magenta triangles; EP as blue circles; EP with the $2^{\text{nd}}$ order correction in (8) as green diamonds. For 20 runs each, the colour intensities correspond to the frequency of reaching different estimates. A Monte Carlo estimate of the true $\ln Z$, as found by parallel tempering with thermodynamic integration, is shown as a line with two-standard deviation error bars.

where $\mu_{x_*} = \mathbf{k}_*^\top \mathbf{K}^{-1}\mathbf{m}$ and variance $s_{x_*}^2 = k_* - \mathbf{k}_*^\top (\mathbf{K} + \mathbf{\Lambda}^{-1})^{-1}\mathbf{k}_*$ and $\mathbf{\Lambda} = \text{diag}(\boldsymbol{\lambda})$ denotes the parameters in the Gaussian terms $g_n$. The average in (20) is over a Gaussian $\mathbf{x}$ with $\mathbf{\Sigma}^{-1} = (\mathbf{K} - k_*^{-1}\mathbf{k}_*\mathbf{k}_*^\top)^{-1} + \mathbf{\Lambda}^{-1}$ and $\boldsymbol{\mu} = (x_* - \mu_{x_*})\sigma_*^{-2}\mathbf{\Sigma}\mathbf{K}^{-1}\mathbf{k}_* + \mathbf{m}$. By simplifying the inner expectation over the complex Gaussian variables $\boldsymbol{\eta}$ we obtain

$$p(x_*) = \mathcal{N}(x_*; \mu_{x_*}, s_{x_*}^2) \left[ 1 + \sum_n \sum_{l \geq 3} \frac{c_{ln}}{l!} \left( \frac{1}{\sqrt{S_{nn}}} \right)^l \left\langle h_l \left( \frac{x_n - m_n}{\sqrt{S_{nn}}} \right) \right\rangle_{\mathbf{x} \sim \mathcal{N}(\mathbf{x}; \boldsymbol{\mu}, \mathbf{\Sigma})} + \cdots \right]$$
(21)

where $h_l$ is the $l^{\text{th}}$ Hermite polynomial. The Hermite polynomials are averaged over a Gaussian density where the only occurrence of $x_*$ is through $(x_* - \mu_{x_*})$ in $\boldsymbol{\mu}$, so that the expansion ultimately appears as a polynomial in $x_*$. A correction to the predictive density follows from averaging $t_*(x_*)$ over (21).

# 4 Applications

## 4.1 Mixture of Gaussians

This section illustrates an example where a large first nontrivial correction term in (8) reflects an inaccurate EP approximation. We explain this for a $K$-component Gaussian mixture model.

Consider $N$ observed data points $\boldsymbol{\zeta}_n$ with likelihood terms $f_n(\mathbf{x}) = \sum_\kappa \pi_\kappa \mathcal{N}(\boldsymbol{\zeta}_n; \boldsymbol{\mu}_\kappa, \mathbf{\Gamma}_\kappa^{-1})$, with $n \geq 1$ and with the mixing weights $\pi_\kappa$ forming a probability vector. The latent variables are then $\mathbf{x} = \{\pi_\kappa, \boldsymbol{\mu}_\kappa, \mathbf{\Gamma}_\kappa\}_{\kappa=1}^K$. For our prior on $\mathbf{x}$ we use a Dirichlet distribution and product of Normal-Wisharts densities so that $f_0(\mathbf{x}) = \mathcal{D}(\boldsymbol{\pi}) \prod_\kappa \mathcal{NW}(\boldsymbol{\mu}_\kappa, \mathbf{\Gamma}_\kappa)$. When we multiply the $f_n$ terms we see that intractability for the mixture model arises because the number of terms in the marginal likelihood is $K^N$, rather than because integration is intractable. The computation of lower-order terms in (8) should therefore be immediately feasible. The approximation $q(\mathbf{x})$ and each $g_n(\mathbf{x})$ are chosen to be of the same exponential family form as $f_0(\mathbf{x})$, where we don't require $g_n(\mathbf{x})$ to be normalizable.

For brevity we omit the details of the EP algorithm for this mixture model, and assume here that an EP fixed point has been found, possibly using some damping. Fig. 1 shows various approximations to the log marginal likelihood $\ln Z$ for $\boldsymbol{\zeta}_n$ coming from the *acidity* data set. It is evident that the "true peak" doesn't match the peak obtained by approximate inference, and we will wrongly predict which $K$ maximizes the log marginal likelihood. Without having to resort to Monte Carlo methods, the second order correction for $K = 3$ both corrects our prediction and already confirms that the original approximation might be inadequate.

## 4.2 Gaussian Process Classification

The GP classification model arises when we observe $N$ data points $\boldsymbol{\zeta}_n$ with class labels $y_n \in \{-1, 1\}$, and model $y$ through a latent function $x$ with the GP prior mentioned in sec. 3.2. The

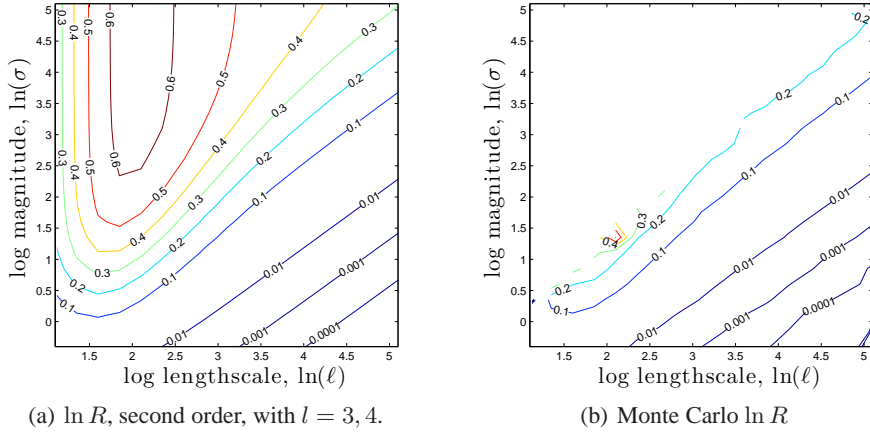

(a) $\ln R$, second order, with $l = 3, 4$.           (b) Monte Carlo $\ln R$

Figure 2: A comparison of a perturbation expansion of (19) against Monte Carlo estimates of the true correction $\ln R$, using the USPS data set from [4].

likelihood terms for $y_n$ are assumed to be $t_n(x_n) = \Phi(y_n x_n)$, where $\Phi(\cdot)$ denotes the cumulative Normal density.

Eq. (19) shows how to compute the cumulant expansion by dovetailing the EP fixed point with the characteristic function of $q_n(x_n)$: From the EP fixed point we have $q(\mathbf{x}) = \mathcal{N}(\mathbf{x}; \mathbf{m}, \mathbf{S})$ and $g_n \propto e^{\gamma_n x_n - \frac{1}{2}\lambda_n x_n}$; consequently the marginal density of $x_n$ in $q(\mathbf{x})/g_n(x_n)$ from (3) is $\mathcal{N}(x_n; \mu, v^2)$, where $v^{-2} = 1/S_{nn} - \lambda_n$ and $\mu = v^{-2}(m_n/S_{nn} - \gamma_n)$. Using (3) again we have

$$q_n(x_n) = \frac{1}{Z_n}\Phi(y_n x_n)\mathcal{N}(x_n; \mu, v^2) . \tag{22}$$

The characteristic function of $q_n(x_n)$ is obtained by the inversion of (12),

$$\chi_n(k) = \left\langle e^{ikx_n} \right\rangle = e^{ik\mu - \frac{1}{2}k^2 v^2}\frac{\Phi(w_k)}{\Phi(w)} , \quad \text{with } w = \frac{y_n\mu}{\sqrt{1+v^2}} \text{ and } w_k = \frac{y_n\mu + ikv^2}{\sqrt{1+v^2}} , \tag{23}$$

with expectations $\langle\cdots\rangle$ being with respect to $q_n(x_n)$. Raw moments are computed through derivatives of the characteristic function, i.e. $\langle x_n^j \rangle = i^{-j}\chi_n^{(j)}(0)$. The cumulants $c_{ln}$ are determined from the derivatives of $\ln\chi_n(k)$ evaluated at zero (or equally from raw moments, e.g. $c_{3n} = 2\langle x_n\rangle^3 - 3\langle x_n\rangle\langle x_n^2\rangle + \langle x_n^3\rangle$), such that

$$c_{3n} = \alpha^3\beta\big[2\beta^2 + 3w\beta + w^2 - 1\big] \tag{24}$$

$$c_{4n} = -\alpha^4\beta\big[6\beta^3 + 12w\beta^2 + 7w^2\beta + w^3 - 4\beta - 3w\big] , \tag{25}$$

where $\alpha = v^2/\sqrt{1+v^2}$ and $\beta = \mathcal{N}(w; 0, 1)/\Phi(w)$.

An extensive MCMC evaluation of EP for GP classification on various data sets was recently given by [4], showing that the log marginal likelihood of the data can be approximated remarkably well. An even more accurate estimation of the approximation error is given by considering the second order correction in (19) (computed here up to $l = 4$). For GPC we generally found that the $l = 3$ term dominates $l = 4$, and we do not include any higher cumulants here. Fig. 2 illustrates the $\ln R$ correction on the binary subproblem of the USPS 3's vs. 5's digits data set, with $N = 767$, as was used by [4]. We used the same kernel $k(\boldsymbol\zeta, \boldsymbol\zeta') = \sigma^2\exp(-\frac{1}{2}\|\boldsymbol\zeta - \boldsymbol\zeta'\|^2/\ell^2)$ as [4], and evaluated (19) on a similar grid of $\ln\ell$ and $\ln\sigma$ values. For the same grid values we obtained Monte Carlo estimates of $\ln Z$, and hence $\ln R$. They are plotted in fig. 2(b) for the cases where they estimate $\ln Z$ to sufficient accuracy (up to four decimal places) to obtain a smoothly varying plot of $\ln R$.[3] The correction from (19), as computed here, is $\mathcal{O}(N^2)$, and compares favourably to $\mathcal{O}(N^3)$ complexity of EP for GPC.

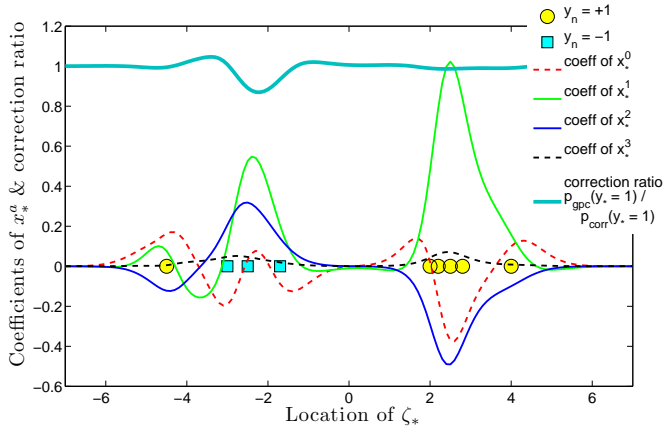

Figure 3: The initial coefficients of the polynomial in $x_*$, as they ultimately appear in the first nontrivial correction term in (21). Cumulants $l = 3$ and $l = 4$ were used. The coefficients are shown for test points $\zeta_*$ after observing data points $\zeta_n$. The ratio between the standard and ($1^{\text{st}}$ order) corrected GP classification predictive density is also illustrated.

In fig. 3 we show the coefficients of the polynomial corrections (21) in powers of $x_*$ to the predictive density $p(x_*)$, using $3^{\text{rd}}$ and $4^{\text{th}}$ cumulants. The small corrections arise as whenever terms $y_n m_n$ are positive and large compared to the posterior variance, non-Gaussian terms $f_n(\mathbf{x}) = t_n(x_n) \approx 1$ for almost all values of $x_n$ which have significant probability under the Gaussian distribution that is proportional to $q(\mathbf{x})/g_n(x_n)$. For these terms $q_n(x)$ is therefore *almost* Gaussian and higher cumulants are small. A example where this will no longer be the case is a GP model with $t_n(x_n) = 1$ for $|x_n| < a$ and $t_n(x_n) = 0$ for $|x_n| > a$. This is a regression model $y_n = x_n + \nu_n$ where i.i.d. noise variables $\nu_n$ have uniform distribution and the observed outputs are all zero, i.e. $y_n = 0$. For this case, the exact posterior variance does not shrink to zero even if the number of data points goes to infinity. The EP approximation however has the variance decrease to zero and our corrections increase with sample size.

## 4.3 Ising models

Somewhat surprising (and probably less known) is the fact that EP and our corrections apply well to a fairly limiting case of the GP model where the terms are of the form $t_n(x_n) = e^{\theta_n x_n} (\delta(x_n + 1) + \delta(x_n - 1))$, where $\delta(x)$ is the Dirac distribution. These terms, together with a "Gaussian" $f_0(\mathbf{x}) = \exp[\sum_{i<j} J_{ij} x_i x_j]$ (where we do not assume that the matrix $\mathbf{J}$ is negative definite), makes this GP model an Ising model with binary variables $x_n = \pm 1$. As shown in [8], this model can still be treated with the same type of Gaussian term approximations as ordinary GP models, allowing for surprisingly accurate estimation of the mean *and* covariance. Here we will show the effect of our corrections for toy models, where exact inference is possible by enumeration.

The tilted distributions $q_n(x_n)$ are biased binary distributions with cumulants: $c_{3n} = -2m_n(1 - m_n^2)$, $c_{4n} = -2 + 8m_n^2 - 6m_n^4$, etc. We will consider two different scenarios for random $\boldsymbol{\theta}$ and $\mathbf{J}$

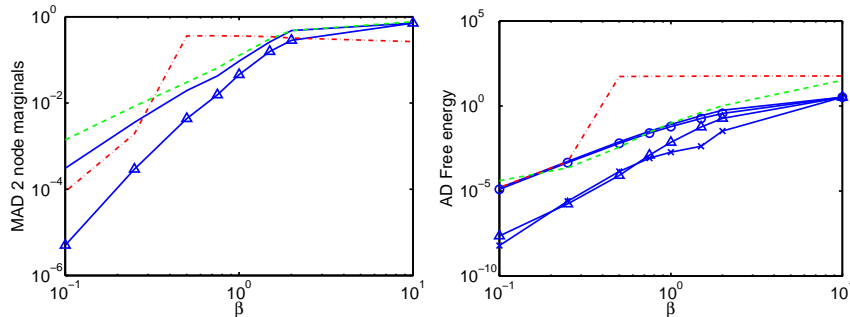

Figure 4: The left plot shows the MAD of the estimated covariance matrix from the exact one for different values of $\beta$ for EP (blue), EP $2^{\text{nd}}$ order $l = 4$ corrections (blue with triangles), Bethe or loopy belief propagation (LBP; dashed green) and Kikuchi or generalized LBP (dash–dotted red). The Bethe and Kikuchi approximations both give covariance estimates for all variable pairs as the model is fully connected. The right plot shows the absolute deviation of $\ln Z$ from the true value using second order perturbations with $l = 3, 4, 5$ ($l = 3$ is the smallest change). The remaining line styles are the same as in the left plot.

described in detail in [8]. In the first scenario, with $N = 10$, the $J_{ij}$'s are generated independently at random according to $J_{ij} = \beta w_{ij}$ and $w_{ij} \sim \mathcal{N}(0, 1)$. For varying $\beta$, the maximum absolute deviation (MAD) of the estimated covariance matrices from the exact one $\max_{i,j} |\Sigma_{ij}^{\text{est}} - \Sigma_{ij}^{\text{exact}}|$ is shown in fig. 4 left. The absolute deviation on the log partition function is shown in fig. 4 right. In the Wainwright-Jordan set-up $N = 16$ nodes are either fully connected or connected to nearest neighbors in a 4–by–4 grid. The external field (observation) strengths $\theta_i$ are drawn from a *uniform* distribution $\theta_i \sim \mathcal{U}[-d_{\text{obs}}, d_{\text{obs}}]$ with $d_{\text{obs}} = 0.25$. Three types of coupling strength statistics are considered: repulsive (anti-ferromagnetic) $J_{ij} \sim \mathcal{U}[-2d_{\text{coup}}, 0]$, mixed $J_{ij} \sim \mathcal{U}[-d_{\text{coup}}, +d_{\text{coup}}]$ and attractive (ferromagnetic) $J_{ij} \sim \mathcal{U}[0, +2d_{\text{coup}}]$. Table 1 gives the MAD of marginals averaged of 100 repetitions. The results for both set-ups give rise to the conclusion that when the EP approximation works well then the correction give an order of magnitude of improvement. In the opposite situation, the correction might worsen the results.

Table 1: Average MAD of marginals in a Wainwright-Jordan set-up, comparing loopy belief propagation (LBP), log-determinant relaxation (LD), EP, EP with $l = 5$ correction (EP+), and EP with only one spanning tree approximating term (EP tree).

| Problem type | | | Method | | | | |
|---|---|---|---|---|---|---|---|
| Graph | Coupling | $d_{\text{coup}}$ | LBP | LD | EP | EP+ | EP tree |
| Full | Repulsive | 0.25 | 0.037 | 0.020 | 0.003 | 0.00058487 | 0.0017 |
| | Repulsive | 0.50 | 0.071 | 0.018 | 0.031 | 0.0157 | 0.0143 |
| | Mixed | 0.25 | 0.004 | 0.020 | 0.002 | 0.00042727 | 0.0013 |
| | Mixed | 0.50 | 0.055 | 0.021 | 0.022 | 0.0159 | 0.0151 |
| | Attractive | 0.06 | 0.024 | 0.027 | 0.004 | 0.0023 | 0.0025 |
| | Attractive | 0.12 | 0.435 | 0.033 | 0.117 | 0.1066 | 0.0211 |
| Grid | Repulsive | 1.0 | 0.294 | 0.047 | 0.153 | 0.1693 | 0.0031 |
| | Repulsive | 2.0 | 0.342 | 0.041 | 0.198 | 0.4244 | 0.0021 |
| | Mixed | 1.0 | 0.014 | 0.016 | 0.011 | 0.0122 | 0.0018 |
| | Mixed | 2.0 | 0.095 | 0.038 | 0.082 | 0.0984 | 0.0068 |
| | Attractive | 1.0 | 0.440 | 0.047 | 0.125 | 0.1759 | 0.0028 |
| | Attractive | 2.0 | 0.520 | 0.042 | 0.177 | 0.4730 | 0.0002 |

## 5   Outlook

We expect that it will be possible to develop similar corrections to other approximate inference methods, such as the variational approach or the "power EP" approximations which interpolate between the variational method and EP. This may help the user to decide which approximation is more accurate for a given problem. We will also attempt an analysis of the scaling of higher order terms in these expansions to see if they are asymptotic or have a finite radius of convergence.

**References**

[1] H. Attias. A variational Bayesian framework for graphical models. In *Advances in Neural Information Processing Systems 12*, 2000.

[2] M. Chertkov and V. Y. Chernyak. Loop series for discrete statistical models on graphs. *Journal of Statistical Mechanics: Theory and Experiment*, page P06009, 2006.

[3] S. Ikeda, T. Tanaka, and S. Amari. Information geometry of turbo and low-density parity-check codes. *IEEE Transactions on Information Theory*, 50(6):1097, 2004.

[4] M. Kuss and C. E. Rasmussen. Assessing approximate inference for binary Gaussian process classification. *Journal of Machine Learning Research*, 6:1679–1704, 2005.

[5] T. P. Minka. Expectation propagation for approximate Bayesian inference. In *UAI 2001*, pages 362–369, 2001.

[6] T. P. Minka. Divergence measures and message passing. Technical Report MSR-TR-2005-173, Microsoft Research, Cambridge, UK, 2005.

[7] T.P. Minka. The EP energy function and minimization schemes. Technical report, 2001.

[8] M. Opper and O. Winther. Expectation consistent approximate inference. *Journal of Machine Learning Research*, 6:2177–2204, 2005.

[9] E. Sudderth, M. Wainwright, and A. Willsky. Loop series and Bethe variational bounds in attractive graphical models. In *Advances in Neural Information Processing Systems 20*, pages 1425–1432. 2008.

## Footnotes

[1]The definition of partition functions $Z_n$ is slightly different from previous works.

[2]The terms in the expansion might be organised in *Feynman graphs*, where "self interaction" loops are absent.

[3]The Monte Carlo estimates in [4] are accurate enough for showing EP's close approximation to $\ln Z$, but not enough to make any quantified statement about $\ln R$.
